# Salient Boundary Detection using Ratio Contour

**Song Wang, Toshiro Kubota**
Dept. Computer Science & Engineering
University of South Carolina
Columbia, SC 29208
{songwang|kubota}@cse.sc.edu

**Jeffrey Mark Siskind**
School Electrical & Comput. Engr.
Purdue University
West Lafayette, IN 47906
qobi@purdue.edu

## Abstract

This paper presents a novel graph-theoretic approach, named ratio contour, to extract perceptually salient boundaries from a set of noisy boundary fragments detected in real images. The boundary saliency is defined using the Gestalt laws of closure, proximity, and continuity. This paper first constructs an undirected graph with two different sets of edges: solid edges and dashed edges. The weights of solid and dashed edges measure the local saliency in and between boundary fragments, respectively. Then the most salient boundary is detected by searching for an optimal cycle in this graph with minimum average weight. The proposed approach guarantees the global optimality without introducing any biases related to region area or boundary length. We collect a variety of images for testing the proposed approach with encouraging results.

## 1 Introduction

Human vision and neural systems possess very strong capabilities of identifying salient structures from various images. Implementing such capabilities on a computer is an important but extremely challenging problem for artificial intelligence, computer vision, and machine learning. The main challenges come from two closely related aspects: (a) the definition of the structural saliency, and (b) the design of efficient algorithms for finding the salient structures. On one hand, we expect very comprehensive and advanced definitions of the saliency so that it models accurately the human perceptual and visual process. On the other hand, we expect simple definitions of saliency so that the global optimum can be found in polynomial time.

Previous methods for salient-structure detection can be grouped into two classes. The first class of methods aims to directly group or segment all the image pixels into some disjoint regions, which are expected to coincide with the underlying salient structures. Earlier efforts include the region-merging/splitting methods, watershed methods, and the active-contour-like methods. Those methods usually have difficulties in finding the globally optimal boundaries in terms of the selected saliency definitions. Recently we have witnessed some advanced methods, like ratio region [5], minimum cut[17], normalized cut [14], globally optimal region/cycle [9], and ratio cut [15], which aim to produce globally optimal boundaries. However, those pixel-grouping based methods usually have difficulties in effectively incorporating perceptual rules, such as boundary smoothness, into their saliency definitions.

Instead of operating directly on the image pixels, another class of methods is designed based on some pre-extracted boundary fragments (or for brevity, *fragments*) [1], which can be obtained using some standard edge-detection methods like Canny detectors. As shown in Fig. 1(a), although those fragments are disconnected and contain serious noise, they provide abundant information on boundary length, tangent directions, and curvatures, which can greatly facilitate the incorporation of advanced perceptual rules like boundary smoothness. Shashua and Ullman [13] presents a parallel network model for detecting salient boundary based on fragment proximity, boundary length, and boundary smoothness. Recent development in this class includes Alter and Basri [2], Jacobs [8], Sarkar and Boyer [12], Guy and Medioni [7], Williams and Thornber [16, 11], and Amir and Lindenbaum [3]. However, many of them still have difficulty in finding the closed boundaries in a sense of global optimality with respect to the given boundary-saliency measure. Elder and Zucker [6] use the shortest-path algorithm to connect fragments to form salient closed boundaries. However, the results have a bias to produce boundaries with shorter length.

This paper presents a new graph based approach to extract salient closed boundaries from a set of fragments detected from real images. This approach seeks a good balance between the complexity of the saliency definition and the complexity of the optimization algorithm. The boundary saliency is based on the well-known Gestalt laws of closure, proximity, and continuity. To avoid the various biases as in Elder and Zucker [6], this paper defines the boundary saliency as the average saliency along the whole boundary. We finally formulate the salient-boundary detection problem into a problem for finding an optimal cycle in an undirected graph. We show this problem is of polynomial time-complexity and give an algorithm to solve it. The proposed algorithm is then tested on a variety of real images.

## 2   Problem Formulation

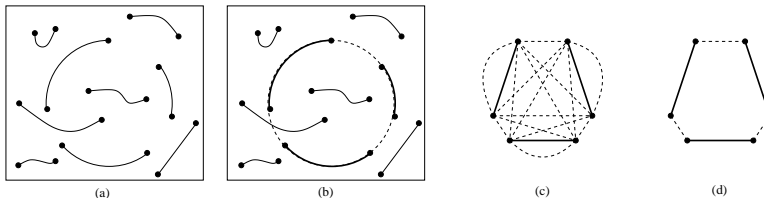

(a)                    (b)                    (c)                    (d)

Figure 1: An illustration of detecting salient boundaries from some fragments. (a) Boundary fragments, (b) salient boundary by connecting some fragments with dashed curves, (c) a solid-dashed graph, and (d) an alternate cycle in (c).

The basic primitives in the ratio-contour approach are a set of noisy (boundary) fragments extracted by edge detection. For simplicity, here we assume each detected fragment is a continuous open curve segment with two endpoints, as shown in Fig. 1(a). Our goal is to identify and connect a subset of fragments to form the most salient structural boundary as shown in Fig. 1(b). In this paper, we measure the boundary saliency using simple Gestalt laws of closure, proximity, and continuity. The closure means that the salient boundary must be a closed contour. The proximity implies that we desire relatively small gaps (dashed curves in Fig. 1(b)) in connecting the fragments. The continuity indicates that the resulting contour should be continuous and sufficiently smooth.

Let the parametric form of a boundary $B$ be $\mathbf{v}(t), 0 \le t \le 1$. We have $\mathbf{v}(0) = \mathbf{v}(1)$ as the boundary is closed. Considering the boundary proximity and the continuity, we define its

cost, which is negatively related to the boundary saliency, as

$$R(B) \triangleq \frac{T(B)}{L(B)} = \frac{\int_B [\sigma(t) + \lambda \cdot \kappa^2(t)]dt}{\int_B dt}, \tag{1}$$

where $\sigma(t) = 1$ if $\mathbf{v}(t)$ is in the gap and $\sigma(t) = 0$, otherwise. $\kappa(t)$ is the curvature at $\mathbf{v}(t)$. We can see that the un-normalized cost $T(B)$ combines the total gap-length and curvature along the boundary $B$ and has bias to produce a short boundary. The issue is addressed in (1) through normalizing $T(B)$ by the boundary length $L(B)$. The most salient boundary $B$ is then the one with the minimum cost $R(B)$. The parameter $\lambda > 0$ is set to balance the weight between proximity and continuity.

We can formulate the above cost into an undirected graph $G = (V, E)$ with vertices $V = \{v_1, v_2, \cdots, v_n\}$ and edges $E = \{e_1, e_2, \cdots, e_m\}$. A unique vertex is constructed from each fragment endpoint. Two different kinds of edges, *solid* edges and *dashed* edges, are constructed between vertices. (a) If $v_i$ and $v_j$ correspond to the two endpoints of the same fragment, we construct a solid edge between $v_i$ and $v_j$ to model this fragment. (b) Between each possible vertex pair $v_i$ and $v_j$, we construct a dashed edge to model the gap or a *virtual fragment* (dashed curves in Fig. 1(b)). An example is shown in Fig. 1(c), which is made up of 3 solid edges for three fragments and all 15 possible dashed edges. For clarity, sometimes we call the boundary fragment a *real fragment* when both real and virtual fragments are involved.

The constructed graph always has even number of vertices, as each real fragment has two endpoints. More interestingly, no two solid edges are incident from the same vertex and each vertex has exactly one incident solid edge. We name such a graph an (undirected) *solid-dashed* graph. We further define an *alternate cycle* in a solid-dashed graph as a simple cycle that traverses the solid edges and dashed edges alternately. Examples of a solid-dashed graph and an alternate cycle are given in Fig. 1(c) and (d), respectively. Since a boundary always traverses real fragments and virtual fragments alternately, it can be described by an alternate cycle. Note that not all the cycles in a solid-dashed graph are alternate cycles, because two adjacent dashed edges can appear sequentially in the same cycle.

According to the cost function (1), we define a weight function $w(e)$ and a length function $l(e)$ for each edge $e$. For convenience, we define $B(e)$ as a function that gives the (real or virtual) fragment corresponding to an edge $e$. Then the weight $w(e) \triangleq T(B(e)) = \int_{B(e)} [\sigma(t) + \lambda \cdot \kappa^2(t)]dt$ is the un-normalized cost on $B(e)$. The edge length $l(e)$ is defined as the length of $B(e)$. We can see that the most salient boundary with minimum cost (1) corresponds to an alternate cycle $C$ with minimum cycle ratio

$$CR(C) = \frac{\sum_{e \in C} w(e)}{\sum_{e \in C} l(e)}.$$

Fragments extracted from real images usually contain noise, intersections, and even some closed curves, which cause difficulties in estimating the curve length, curvature, and therefore, the edge weight and length. We will describe a spline-based method to address this problem in Section 4. In the following, we first present a polynomial-time algorithm to identify the alternate cycle with the minimum cycle ratio $CR(C)$.

## 3  Ratio-Contour Algorithm

For simplicity, we denote the alternate cycle with minimum cycle ratio as MRA (Minimum Ratio Alternate) cycle. In this section, we introduce a graph algorithm for finding the MRA cycle in polynomial time. This algorithm consists of three reductions. (a) Both the weight

and edge length of the solid edges can be set to zero by merging them into the weight and length of their adjacent dashed edges, without changing the underlying MRA. (b) The problem of finding an MRA cycle can be reduced to a problem of detecting a negative-weight alternate (NWA) cycle in the same graph. (c) Finding NWA cycles in a solid-dashed graph with zero solid-edge weights and zero solid-edge lengths can be reduced to finding a minimum-weight perfect matching (MWPM) in the same graph. Finding MWPM has been shown to be of polynomial-time complexity with various efficient algorithms available.

## 3.1 Setting Zero-Weight and Zero-Length to Solid Edges

As illustrated in Fig. 2(a) and (b), each solid edge $e$ can only be adjacent to a set of dashed edges, say $\{e_1, e_2, \cdots, e_K\}$, in a solid-dashed graph, and no two solid edges are adjacent to each other. Therefore, we can directly merge the solid-edge weight and length to its adjacent dashed edges by

$$
\begin{cases}
w(e_k) \leftarrow w(e_k) + \frac{w(e)}{N_k} \\
l(e_k) \leftarrow l(e_k) + \frac{l(e)}{N_k}, & k = 1, 2, \cdots K,
\end{cases}
$$

where $N_k = 2$ if $e_k$ shares one vertex with $e$ as in Fig. 2(a) and $N_k = 1$ if $e_k$ shares both vertices with $e$ as in Fig. 2(b). Then we reset the weight and length of this solid edge to zero, i.e., $w(e) = 0, l(e) = 0$. This merging process is performed on all solid edges. While solid and dashed edges are traversed alternately in an alternate cycle, it is not difficult to achieve the following conclusion.

**Lemma 3.1** *The above processing of edge weights and edge-lengths does not change the cycle ratio of any alternate cycles.*

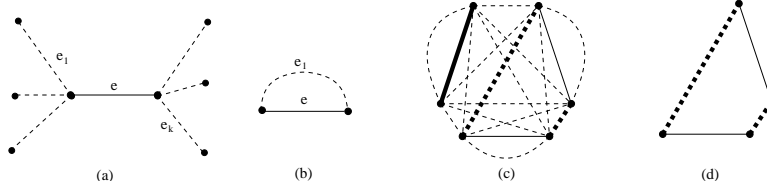

Figure 2: An illustration of reductions in ratio-contour algorithm. (a) Merging the weight and length of a solid edge to its adjacent dashed edges. (b) A special case for weight merging. (c) A perfect matching in a solid-dashed graph. (d) Derived cycle from the perfect matching shown in (c).

## 3.2 Reducing to Negative-Alternate-Cycle Detection

The following lemma claims that MRA cycles are invariant to some more general linear edge-weight transforms.

**Lemma 3.2** *The MRA cycle in a solid-dashed graph $G = (V, E)$ is invariant to the following linear transform on the edge weight*

$$
w(e) \leftarrow w(e) - b \cdot l(e), \forall e \in E. \tag{2}
$$

The proof for this lemma is similar to the one we gave for general ratio-cycle detection problem [15]. Notice that all the edge lengths are non-negative. There always exists an optimal $b = b^*$ so that after weight transform (2), the MRA cycle has the cycle ratio of zero. In this case, the MRA cycle is the same as the cycle with total edge weight of zero.

The detection of the optimal $b^*$ and the MRA cycle can then be reduced into a problem of finding the NWA cycle (negative weight alternate cycle). Basically, if we can detect an NWA cycle after the edge weight transform (2), we know $b > b^*$. Otherwise, we know that $b \leq b^*$. With an NWA cycle detection algorithm, we can use binary or sequential search to locate the optimal $b^*$ and the desired MRA cycle. This search process is polynomial if all the edge weight are integers [15]. In addition, with the first reduction mentioned in Section 3.1, it is easy to see that the linear transform (2) always preserves zero weight and zero length for all solid edges in this search process.

### 3.3  Reducing to Minimum Weight Perfect Matching

The problem of detecting an NWA cycle in a solid-dashed graph can be reduced to a problem of finding a minimum weight perfect matching (MWPM) in the same graph. A perfect matching in $G$ denotes a subgraph that contains all the vertices in $G$ while each vertex only has one incident edge. An example is shown in Fig. 2(c), where three thick edges together with their vertices form a perfect matching. The MWPM is the perfect matching with minimum total edge weight. As all the solid edges form a trivial perfect matching with total weight zero, the MWPM in our solid-dashed graph should have non-positive total weight.

We can construct a set of cycles from a perfect matching $P$ by (a) removing from $P$ all the solid edges and their endpoints, and (b) adding to $P$ any solid edges in the solid-dashed graph $G$ whose two endpoints are still in $P$ after the removal in (a). The remaining subgraph must consist of a set of cycles because each remaining vertex has two incident edges: one is solid and the other one is dashed. This also confirms that all the resulting cycles are alternate cycles. An example of this reduction is shown in Fig. 2(d), which is constructed from (c). As all the solid edges have zero weight and zero length, it is not difficult to see that the total weight of the perfect matching is the same as the total weight of the resulting cycles. Therefore, the NWA detection problem is reduced into a problem of finding a perfect matching with negative total weight. This is the same as the problem of finding the MWPM, which is of polynomial-time complexity [1].

## 4  Edge-Weight and Edge-Length Functions

We need to estimate the curvature and length of both real and virtual fragments for defining $w(e)$ and $l(e)$ of solid and dashed edges. To deal with the noise and aliasing in detected fragments, we impose a pre-smoothing process on those fragments. In this paper, we approximate a fragment by a set of quadratic splines with the parametric form

$$\begin{pmatrix} x_i(t_i) \\ y_i(t_i) \end{pmatrix} = \begin{pmatrix} x_i \\ y_i \end{pmatrix} + \begin{pmatrix} A_i & B_i \\ C_i & D_i \end{pmatrix} \begin{pmatrix} t_i^2 \\ t_i \end{pmatrix},$$

where $0 \leq t_i \leq 1$ is the parameter for the spline. We developed an iterative algorithm [10] to estimate the optimal parameters $x_i$, $y_i$, $A_i$, $B_i$, $C_i$, and $D_i$ minimizing a comprehensive cost function that measures smoothness, under the constraint of $C^0$ and $C^1$ continuities across the fragment. An example is illustrated in Fig. 3 where solid curves in (a) and (b) are fragments before and after smoothing. More discussion and analysis on this curve-smoothing method can be found in our previous work [10].

With the parametric form of quadratic splines, the total length and the curvature along a real fragment can be computed by summing over each spline its length and its total curvature as

$$l_i = \int_0^1 \sqrt{(2A_it + B_i)^2 + (2C_it + D_i)^2}\,dt,$$

$$\int_0^1 \kappa_i^2(t)dt = \int_0^1 \frac{4(A_iD_i - B_iC_i)^2}{[(2A_it + B_i)^2 + (2C_it + D_i)^2]^3}\,dt,$$

where $l_i$ is the length and $\kappa_i(t)$ is the curvature function of the $i$th spline.

However, estimating these quantities for a virtual fragment is not trivial, as no information is given on how the virtual fragment should look like. We take the following approach to compute the dashed-edge weight. First, a pair of endpoints involved in forming a particular dashed edge is connected with a straight line. Then a new curve segment is constructed by connecting this straight line and adjacent fragments. The smoothing process described above is applied to this new curve segment. In this smoothed curve segment, the virtual fragment is then the part corresponding to the straight line before the smoothing. The dashed curve in Fig. 3(b) shows a resulting virtual fragment used for estimating curvature, length, and finally edge weight.

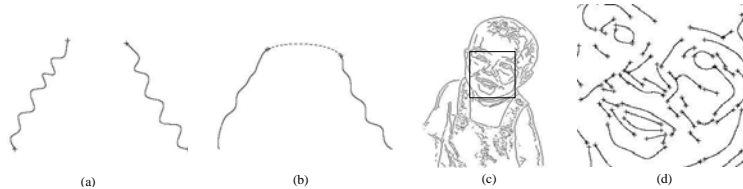

(a)                          (b)                          (c)                          (d)

Figure 3: An illustration of the edge weight estimation process. (a) Two noisy fragments. (b) Smoothed real fragments and an estimated virtual fragment. (c) Fragments obtained by Canny detector. (d) Smoothed fragments after breaking undesired connections, corresponding to the portion of the box in (c). Crossings specify the endpoints and breaking points.

In real implementation, another issue is that the detected fragments using edge detectors may not be disjoint open curves as assumed in Section 2. It is common that some fragments are connected in the form of intersections, attachments, and even undesired closure, as shown in Fig. 4. Therefore, we need to break those connections to construct the graph model. First, we identify the intersection points and split them to get multiple open fragments. An example is shown in Fig. 4(a) and (d), where an intersection point is broken into three endpoints. In the constructed graph, they ($u_1$, $u_2$, and $u_3$) are connected by dashed edges with zero weight and zero length. Attachment specifies the case where two fragments are undesirably connected into a single fragment as shown in Fig. 4(b). This greatly hurts the reliability of salient boundary detection as those attached fragments may exclude many desired dashed edges from the graph. We alleviate this problem by splitting all the fragments at their high-curvature points, as illustrated in Figs. 4(b) and (e). Similarly, we can break closed fragments into open fragments at high-curvature points, as shown in Fig. 4(c) and (f). Note that the identification of high-curvature points requires the smoothing of the noisy fragments. We apply the same smoothing technique described above to each fragment for this purpose. Figures 3(c) and (d) show an example of dealing with the above special cases.

## 5 Experiments and Discussion

In this section, we test the proposed ratio-contour algorithm to extract the salient boundaries from real images. For initial fragment detection, we use the standard Canny edge detector in the Matlab software with its default threshold settings. We also adopt the `Blossom4` implementation [4] of the minimum-weight perfect matching.

One problem in the implementation is the construction of dashed edges, which may be of a very large number ($O(n^2)$) if we connect every two possible vertices. In this paper, we constrain the proximity to reduce the number of dashed edges. In the implementation, for each vertex, we only keep certain number of incident dashed edges with smallest length.

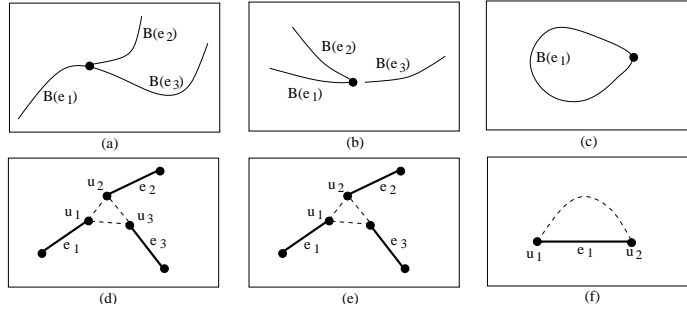

Figure 4: An illustration of fragment identification and graph construction in some special cases. (a), (b), and (c) show the detected fragments with intersections, attachments, and closures. (d), (e), and (f) are the constructed graphs from (a), (b), and (c), respectively.

This number is uniformly set to 20 in all experiments. Meanwhile, we set the parameter $\lambda = 50$ in the edge-weight definition. Figure 5 shows salient boundaries detected from seven real images, together with the initial fragments from Canny detector. It can be seen that the proposed method integrates well the Gestalt laws of proximity, continuity, and closure.

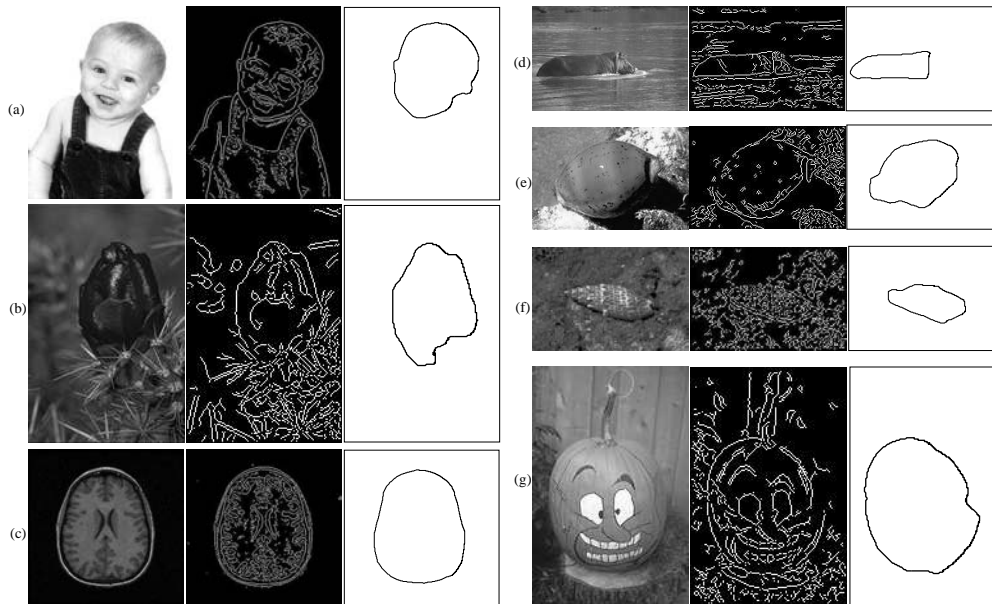

Figure 5: Salient boundaries detected from some real images using the proposed ratio-contour algorithm. Each subfigure from (a) to (g) contains three images, left: original images, middle: Canny detection results, and right: the detected most salient boundaries.

## 6 Conclusions

We have presented a novel graph-theoretic approach, named ratio contour, for extracting perceptually salient boundaries from a set of noisy boundary fragments detected in real images. The approach guarantees the global optimality without introducing any biases re-

lated to region area or boundary length, and exhibits promising performance in extracting salient objects from real cluttered images. One potential extension of this research is to extract multiple salient objects that are overlapped or share part of boundaries by performing ratio-contour algorithm iteratively. We are currently investigating this extension and plan on reporting the result in the future.

## Acknowledgements

The authors would like to thank David Jacobs and anonymous reviewers for important comments. This work was funded, in part, by National Science Foundation grant EIA-0312861, and the USC SOM-COEIT research development fund.

## Footnotes

[1]Most literatures use the terminology *edge* instead of *fragment*. However, in this paper *edge* has other specified meaning in a graph model.

## References

[1] R. K. Ahuja, T. L. Magnanti, and J. B. Orlin. *Network Flows: Theory, Algorithms, & Applications*. Prentice Hall, Englewood Cliffs, 1993.

[2] T. Alter and R. Basri. Extracting salient contours from images: An analysis of the saliency network. In *IEEE Conference on Computer Vision and Pattern Recognition*, pages 13–20, 1996.

[3] A. Amir and M. Lindenbaum. A generic grouping algorithm and its quantitative analysis. *IEEE Transactions on Pattern Analysis and Machine Intelligence*, 20(2):168–185, 1998.

[4] W. Cook and A. Rohe. Computing minimum-weight perfect matchings. http://www.or.unibonn.de/home/rohe/matching.html, Aug. 1998.

[5] I. Cox, S. B. Rao, and Y. Zhong. Ratio regions: A technique for image segmentation. In *International Conference on Pattern Recognition*, pages 557–564, 1996.

[6] J. Elder and S. Zucker. Computing contour closure. In *European Conference on Computer Vision*, pages 399–412, 1996.

[7] G. Guy and G. Medioni. Inferring global perceptual contours from local features. *International Journal of Computer Vision*, 20(1):113–133, 1996.

[8] D. Jacobs. Robust and efficient detection of convex groups. *IEEE Transactions on Pattern Analysis and Machine Intelligence*, 18(1):23–37, 1996.

[9] I. H. Jermyn and H. Ishikawa. Globally optimal regions and boundaries as minimum ratio cycles. *IEEE Transactions on Pattern Analysis and Machine Intelligence*, 23(10):1075–1088, 2001.

[10] T. Kubota. Contextual and non-combinatorial approach to feature extraction. In *Int'l Workshop on EMMCVPR*, pages 467–482, 2003.

[11] S. Mahamud, L. R. Williams, K. K. Thornber, and K. Xu. Segmentation of multiple salient closed contours from real images. *IEEE Transactions on Pattern Analysis and Machine Intelligence*, 25(4):433–444, 2003.

[12] S. Sarkar and K. Boyer. Quantitative measures of change bvased on feature organization: Eigenvalues and eigenvectors. In *IEEE Conference on Computer Vision and Pattern Recognition*, pages 478–483, 1996.

[13] A. Shashua and S. Ullman. Structural saliency: The detection of globallly salient structures using a locally connected network. In *International Conference on Computer Vision*, pages 321–327, 1988.

[14] J. Shi and J. Malik. Normalized cuts and image segmentation. *IEEE Transactions on Pattern Analysis and Machine Intelligence*, 22(8):888–905, 2000.

[15] S. Wang and J. M. Siskind. Image segmentation with ratio cut. *IEEE Transactions on Pattern Analysis and Machine Intelligence*, 25(6):675–690, 2003.

[16] L. Williams and K. K. Thornber. A comparison measures for detecting natural shapes in cluttered background. *International Journal of Computer Vision*, 34(2/3):81–96, 2000.

[17] Z. Wu and R. Leahy. An optimal graph theoretic approach to data clustering: Theory and its application to image segmentation. *IEEE Transactions on Pattern Analysis and Machine Intelligence*, 15(11):1101–1113, 1993.
